# Sparse Gaussian Processes using Pseudo-inputs

**Edward Snelson**          **Zoubin Ghahramani**

Gatsby Computational Neuroscience Unit
University College London
17 Queen Square, London WC1N 3AR, UK
{snelson,zoubin}@gatsby.ucl.ac.uk

## Abstract

We present a new Gaussian process (GP) regression model whose co-variance is parameterized by the the locations of $M$ pseudo-input points, which we learn by a gradient based optimization. We take $M \ll N$, where $N$ is the number of real data points, and hence obtain a sparse regression method which has $\mathcal{O}(M^2N)$ training cost and $\mathcal{O}(M^2)$ prediction cost per test case. We also find hyperparameters of the covariance function in the same joint optimization. The method can be viewed as a Bayesian regression model with particular input dependent noise. The method turns out to be closely related to several other sparse GP approaches, and we discuss the relation in detail. We finally demonstrate its performance on some large data sets, and make a direct comparison to other sparse GP methods. We show that our method can match full GP performance with small $M$, i.e. very sparse solutions, and it significantly outperforms other approaches in this regime.

## 1   Introduction

The Gaussian process (GP) is a popular and elegant method for Bayesian non-linear non-parametric regression and classification. Unfortunately its non-parametric nature causes computational problems for large data sets, due to an unfavourable $N^3$ scaling for training, where $N$ is the number of data points. In recent years there have been many attempts to make sparse approximations to the full GP in order to bring this scaling down to $M^2N$ where $M \ll N$ [1, 2, 3, 4, 5, 6, 7, 8, 9]. Most of these methods involve selecting a subset of the training points of size $M$ (active set) on which to base computation. A typical way of choosing such a subset is through some sort of information criterion. For example, Seeger et al. [7] employ a very fast approximate information gain criterion, which they use to greedily select points into the active set.

A major common problem to these methods is that they lack a reliable way of learning kernel hyperparameters, because the active set selection interferes with this learning procedure. Seeger et al. [7] construct an approximation to the full GP marginal likelihood, which they try to maximize to find the hyperparameters. However, as the authors state, they have persistent difficulty in practically doing this through gradient ascent. The reason for this is that reselecting the active set causes non-smooth fluctuations in the marginal likelihood

and its gradients, meaning that they cannot get smooth convergence. Therefore the speed of active set selection is somewhat undermined by the difficulty of selecting hyperparameters. Inappropriately learned hyperparameters will adversely affect the quality of solution, especially if one is trying to use them for *automatic relevance determination* (ARD) [10].

In this paper we circumvent this problem by constructing a GP regression model that enables us to find active set point locations and hyperparameters in one smooth joint optimization. The covariance function of our GP is parameterized by the locations of pseudo-inputs — an active set not constrained to be a subset of the data, found by a continuous optimization. This is a further major advantage, since we can improve the quality of our fit by the fine tuning of their precise locations.

Our model is closely related to several sparse GP approximations, in particular Seeger's method of *projected latent variables* (PLV) [7, 8]. We discuss these relations in section 3. In principle we could also apply our technique of moving active set points off data points to approximations such as PLV. However we empirically demonstrate that a crucial difference between PLV and our method (SPGP) prevents this idea from working for PLV.

## 1.1   Gaussian processes for regression

We provide here a concise summary of GPs for regression, but see [11, 12, 13, 10] for more detailed reviews. We have a data set $\mathcal{D}$ consisting of $N$ input vectors $\mathbf{X} = \{\mathbf{x}_n\}_{n=1}^N$ of dimension $D$ and corresponding real valued targets $\mathbf{y} = \{y_n\}_{n=1}^N$. We place a zero mean Gaussian process prior on the underlying latent function $f(x)$ that we are trying to model. We therefore have a multivariate Gaussian distribution on any finite subset of latent variables; in particular, at $\mathbf{X}$: $p(\mathbf{f}|\mathbf{X}) = \mathcal{N}(\mathbf{f}|\mathbf{0}, \mathbf{K}_N)$, where $\mathcal{N}(\mathbf{f}|\mathbf{m}, \mathbf{V})$ is a Gaussian distribution with mean $\mathbf{m}$ and covariance $\mathbf{V}$. In a Gaussian process the covariance matrix is constructed from a covariance function, or kernel, $K$ which expresses some prior notion of smoothness of the underlying function: $[\mathbf{K}_N]_{nn'} = K(\mathbf{x}_n, \mathbf{x}_{n'})$. Usually the covariance function depends on a small number of hyperparameters $\boldsymbol{\theta}$, which control these smoothness properties. For our experiments later on we will use the standard Gaussian covariance with ARD hyperparameters:

$$K(\mathbf{x}_n, \mathbf{x}_{n'}) = c \exp\left[-\tfrac{1}{2} \sum_{d=1}^{D} b_d \big(x_n^{(d)} - x_{n'}^{(d)}\big)^2\right], \qquad \boldsymbol{\theta} = \{c, \mathbf{b}\} . \tag{1}$$

In standard GP regression we also assume a Gaussian noise model or likelihood $p(\mathbf{y}|\mathbf{f}) = \mathcal{N}(\mathbf{y}|\mathbf{f}, \sigma^2 \mathbf{I})$. Integrating out the latent function values we obtain the marginal likelihood:

$$p(\mathbf{y}|\mathbf{X}, \boldsymbol{\theta}) = \mathcal{N}(\mathbf{y}|\mathbf{0}, \mathbf{K}_N + \sigma^2 \mathbf{I}) , \tag{2}$$

which is typically used to train the GP by finding a (local) maximum with respect to the hyperparameters $\boldsymbol{\theta}$ and $\sigma^2$.

Prediction is made by considering a new input point $\mathbf{x}$ and conditioning on the observed data and hyperparameters. The distribution of the target value at the new point is then:

$$p(y|\mathbf{x}, \mathcal{D}, \boldsymbol{\theta}) = \mathcal{N}\big(y\big|\mathbf{k}_\mathbf{x}^\top (\mathbf{K}_N + \sigma^2 \mathbf{I})^{-1} \mathbf{y}, \ K_{\mathbf{xx}} - \mathbf{k}_\mathbf{x}^\top (\mathbf{K}_N + \sigma^2 \mathbf{I})^{-1} \mathbf{k}_\mathbf{x} + \sigma^2\big) , \tag{3}$$

where $[\mathbf{k}_\mathbf{x}]_n = K(\mathbf{x}_n, \mathbf{x})$ and $K_{\mathbf{xx}} = K(\mathbf{x}, \mathbf{x})$. The GP is a non-parametric model, because the training data are explicitly required at test time in order to construct the predictive distribution, as is clear from the above expression.

GPs are prohibitive for large data sets because training requires $\mathcal{O}(N^3)$ time due to the inversion of the covariance matrix. Once the inversion is done, prediction is $\mathcal{O}(N)$ for the predictive mean and $\mathcal{O}(N^2)$ for the predictive variance per new test case.

## 2 Sparse Pseudo-input Gaussian processes (SPGPs)

In order to derive a sparse model that is computationally tractable for large data sets, which still preserves the desirable properties of the full GP, we examine in detail the GP predictive distribution (3). Consider the mean and variance of this distribution as functions of $\mathbf{x}$, the new input. Regarding the hyperparameters as known and fixed for now, these functions are effectively parameterized by the locations of the $N$ training input and target pairs, $\mathbf{X}$ and $\mathbf{y}$. In this paper we consider a model with likelihood given by the GP predictive distribution, and parameterized by a *pseudo data set*. The sparsity in the model will arise because we will generally consider a pseudo data set $\bar{\mathcal{D}}$ of size $M < N$: pseudo-inputs $\bar{\mathbf{X}} = \{\bar{\mathbf{x}}_m\}_{m=1}^M$ and pseudo targets $\bar{\mathbf{f}} = \{\bar{f}_m\}_{m=1}^M$. We have denoted the pseudo targets $\bar{\mathbf{f}}$ instead of $\bar{\mathbf{y}}$ because as they are not real observations, it does not make much sense to include a noise variance for them. They are therefore equivalent to the latent function values $\mathbf{f}$. The actual observed target value will of course be assumed noisy as before. These assumptions therefore lead to the following single data point likelihood:

$$p(y|\mathbf{x}, \bar{\mathbf{X}}, \bar{\mathbf{f}}) = \mathcal{N}\big(y\big|\mathbf{k}_{\mathbf{x}}^\top \mathbf{K}_M^{-1}\bar{\mathbf{f}}, \ K_{\mathbf{x}\mathbf{x}} - \mathbf{k}_{\mathbf{x}}^\top \mathbf{K}_M^{-1}\mathbf{k}_{\mathbf{x}} + \sigma^2\big) \,, \tag{4}$$

where $[\mathbf{K}_M]_{mm'} = K(\bar{\mathbf{x}}_m, \bar{\mathbf{x}}_{m'})$ and $[\mathbf{k}_{\mathbf{x}}]_m = K(\bar{\mathbf{x}}_m, \mathbf{x})$, for $m = 1, \dots, M$.

This can be viewed as a standard regression model with a particular form of parameterized mean function and input-dependent noise model. The target data are generated i.i.d. given the inputs, giving the complete data likelihood:

$$p(\mathbf{y}|\mathbf{X}, \bar{\mathbf{X}}, \bar{\mathbf{f}}) = \prod_{n=1}^N p(y_n|\mathbf{x}_n, \bar{\mathbf{X}}, \bar{\mathbf{f}}) = \mathcal{N}(\mathbf{y}|\mathbf{K}_{NM}\mathbf{K}_M^{-1}\bar{\mathbf{f}}, \ \mathbf{\Lambda} + \sigma^2\mathbf{I}) \,, \tag{5}$$

where $\mathbf{\Lambda} = \mathrm{diag}(\boldsymbol{\lambda})$, $\lambda_n = K_{nn} - \mathbf{k}_n^\top \mathbf{K}_M^{-1}\mathbf{k}_n$, and $[\mathbf{K}_{NM}]_{nm} = K(\mathbf{x}_n, \bar{\mathbf{x}}_m)$.

Learning in the model involves finding a suitable setting of the parameters – an appropriate pseudo data set that explains the real data well. However rather than simply maximize the likelihood with respect to $\bar{\mathbf{X}}$ and $\bar{\mathbf{f}}$ it turns out that we can integrate out the pseudo targets $\bar{\mathbf{f}}$. We place a Gaussian prior on the pseudo targets:

$$p(\bar{\mathbf{f}}|\bar{\mathbf{X}}) = \mathcal{N}(\bar{\mathbf{f}}|\mathbf{0}, \mathbf{K}_M) \,. \tag{6}$$

This is a very reasonable prior because we expect the pseudo data to be distributed in a very similar manner to the real data, if they are to model them well. It is not easy to place a prior on the pseudo-inputs and still remain with a tractable model, so we will find these by maximum likelihood (ML). For the moment though, consider the pseudo-inputs as known.

We find the posterior distribution over pseudo targets $\bar{\mathbf{f}}$ using Bayes rule on (5) and (6):

$$p(\bar{\mathbf{f}}|\mathcal{D}, \bar{\mathbf{X}}) = \mathcal{N}\big(\bar{\mathbf{f}}\big|\mathbf{K}_M\mathbf{Q}_M^{-1}\mathbf{K}_{MN}(\mathbf{\Lambda} + \sigma^2\mathbf{I})^{-1}\mathbf{y}, \ \mathbf{K}_M\mathbf{Q}_M^{-1}\mathbf{K}_M\big) \,, \tag{7}$$

where $\mathbf{Q}_M = \mathbf{K}_M + \mathbf{K}_{MN}(\mathbf{\Lambda} + \sigma^2\mathbf{I})^{-1}\mathbf{K}_{NM}$.

Given a new input $\mathbf{x}_*$, the predictive distribution is then obtained by integrating the likelihood (4) with the posterior (7):

$$p(y_*|\mathbf{x}_*, \mathcal{D}, \bar{\mathbf{X}}) = \int d\bar{\mathbf{f}} \ p(y_*|\mathbf{x}_*, \bar{\mathbf{X}}, \bar{\mathbf{f}}) \, p(\bar{\mathbf{f}}|\mathcal{D}, \bar{\mathbf{X}}) = \mathcal{N}(y_*|\mu_*, \sigma_*^2) \,, \tag{8}$$

where
$$\mu_* = \mathbf{k}_*^\top \mathbf{Q}_M^{-1}\mathbf{K}_{MN}(\mathbf{\Lambda} + \sigma^2\mathbf{I})^{-1}\mathbf{y}$$
$$\sigma_*^2 = K_{**} - \mathbf{k}_*^\top(\mathbf{K}_M^{-1} - \mathbf{Q}_M^{-1})\mathbf{k}_* + \sigma^2 \,.$$

Note that inversion of the matrix $\mathbf{\Lambda} + \sigma^2\mathbf{I}$ is not a problem because it is diagonal. The computational cost is dominated by the matrix multiplication $\mathbf{K}_{MN}(\mathbf{\Lambda} + \sigma^2\mathbf{I})^{-1}\mathbf{K}_{NM}$ in the calculation of $\mathbf{Q}_M$ which is $\mathcal{O}(M^2 N)$. After various precomputations, prediction can then be made in $\mathcal{O}(M)$ for the mean and $\mathcal{O}(M^2)$ for the variance per test case.

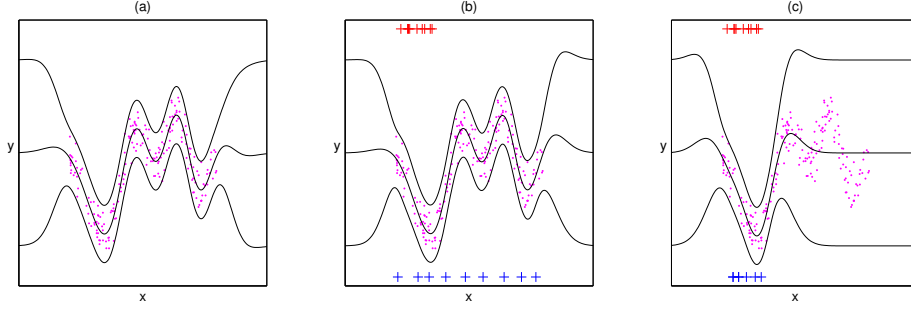

Figure 1: Predictive distributions (mean and two standard deviation lines) for: (a) full GP, (b) SPGP trained using gradient ascent on (9), (c) SPGP trained using gradient ascent on (10). Initial pseudo point positions are shown at the top as red crosses; final pseudo point positions are shown at the bottom as blue crosses (the $y$ location on the plots of these crosses is *not* meaningful).

We are left with the problem of finding the pseudo-input locations $\bar{\mathbf{X}}$ and hyperparameters $\boldsymbol{\Theta} = \{\boldsymbol{\theta}, \sigma^2\}$. We can do this by computing the marginal likelihood from (5) and (6):

$$
\begin{aligned}
p(\mathbf{y}|\mathbf{X}, \bar{\mathbf{X}}, \boldsymbol{\Theta}) &= \int d\bar{\mathbf{f}} \; p(\mathbf{y}|\mathbf{X}, \bar{\mathbf{X}}, \bar{\mathbf{f}}) \, p(\bar{\mathbf{f}}|\bar{\mathbf{X}}) \\
&= \mathcal{N}(\mathbf{y}|\mathbf{0}, \; \mathbf{K}_{NM}\mathbf{K}_M^{-1}\mathbf{K}_{MN} + \boldsymbol{\Lambda} + \sigma^2\mathbf{I}) \; .
\end{aligned}
\tag{9}
$$

The marginal likelihood can then be maximized with respect to all these parameters $\{\bar{\mathbf{X}}, \boldsymbol{\Theta}\}$ by gradient ascent. The details of the gradient calculations are long and tedious and therefore omitted here for brevity. They closely follow the derivations of hyperparameter gradients of Seeger et al. [7] (see also section 3), and as there, can be most efficiently coded with Cholesky factorisations. Note that $\mathbf{K}_M$, $\mathbf{K}_{MN}$ and $\boldsymbol{\Lambda}$ are all functions of the $M$ pseudo-inputs $\bar{\mathbf{X}}$ and $\boldsymbol{\theta}$. The exact form of the gradients will of course depend on the functional form of the covariance function chosen, but our method will apply to any covariance that is differentiable with respect to the input points. It is worth saying that the SPGP can be viewed as a standard GP with a particular non-stationary covariance function parameterized by the pseudo-inputs.

Since we now have $MD + |\boldsymbol{\Theta}|$ parameters to fit, instead of just $|\boldsymbol{\Theta}|$ for the full GP, one may be worried about overfitting. However, consider the case where we let $M = N$ and $\bar{\mathbf{X}} = \mathbf{X}$ – the pseudo-inputs coincide with the real inputs. At this point the marginal likelihood is equal to that of a full GP (2). This is because at this point $\mathbf{K}_{MN} = \mathbf{K}_M = \mathbf{K}_N$ and $\boldsymbol{\Lambda} = \mathbf{0}$. Moreover the predictive distribution (8) also collapses to the full GP predictive distribution (3). These are clearly desirable properties of the model, and they give confidence that a good solution will be found when $M < N$. However it is the case that hyperparameter learning complicates matters, and we discuss this further in section 4.

## 3   Relation to other methods

It turns out that Seeger's method of PLV [7, 8] uses a very similar marginal likelihood approximation and predictive distribution. If you remove $\boldsymbol{\Lambda}$ from all the SPGP equations you get precisely their expressions. In particular the marginal likelihood they use is:

$$
p(\mathbf{y}|\mathbf{X}, \bar{\mathbf{X}}, \boldsymbol{\Theta}) = \mathcal{N}(\mathbf{y}|\mathbf{0}, \; \mathbf{K}_{NM}\mathbf{K}_M^{-1}\mathbf{K}_{MN} + \sigma^2\mathbf{I}) \; ,
\tag{10}
$$

which has also been used elsewhere before [1, 4, 5]. They have derived this expression from a somewhat different route, as a direct approximation to the full GP marginal likelihood.

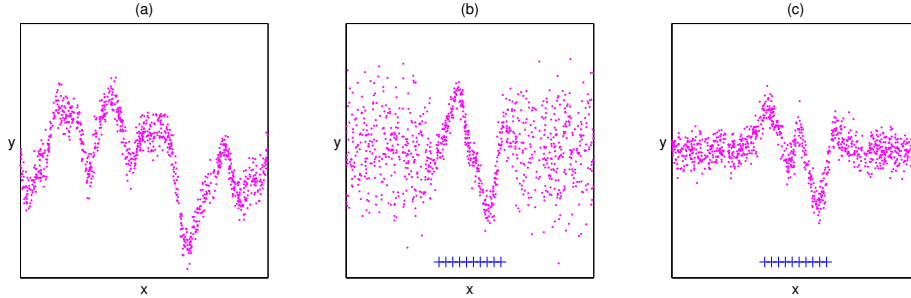

Figure 2: Sample data drawn from the marginal likelihood of: (a) a full GP, (b) SPGP, (c) PLV. For (b) and (c), the blue crosses show the location of the 10 pseudo-input points.

As discussed earlier, the major difference between our method and these other methods, is that they do not use this marginal likelihood to learn locations of active set input points – only the hyperparameters are learnt from (10). This begged the question of what would happen if we tried to use their marginal likelihood approximation (10) instead of (9) to try to learn pseudo-input locations by gradient ascent. We show that the $\Lambda$ that appears in the SPGP marginal likelihood (9) is crucial for finding pseudo-input points by *gradients*.

Figure 1 shows what happens when we try to optimize these two likelihoods using gradient ascent with respect to the pseudo inputs, on a simple 1D data set. Plotted are the predictive distributions, initial and final locations of the pseudo inputs. Hyperparameters were fixed to their true values for this example. The initial pseudo-input locations were chosen adversarially: all towards the left of the input space (red crosses). Using the SPGP likelihood, the pseudo-inputs spread themselves along the extent of the training data, and the predictive distribution matches the full GP very closely (Figure 1(b)). Using the PLV likelihood, the points begin to spread, but very quickly become stuck as the gradient pushing the points towards the right becomes tiny (Figure 1(c)).

Figure 2 compares data sampled from the marginal likelihoods (9) and (10), given a particular setting of the hyperparameters and a small number of pseudo-input points. The major difference between the two is that the SPGP likelihood has a constant marginal variance of $K_{nn} + \sigma^2$, whereas the PLV decreases to $\sigma^2$ away from the pseudo-inputs. Alternatively, the *noise* component of the PLV likelihood is a constant $\sigma^2$, whereas the SPGP noise grows to $K_{nn} + \sigma^2$ away from the pseudo-inputs. If one is in the situation of Figure 1(c), under the SPGP likelihood, moving the rightmost pseudo-input slightly to the right will immediately start to reduce the noise in this region from $K_{nn} + \sigma^2$ towards $\sigma^2$. Hence there will be a strong gradient pulling it to the right. With the PLV likelihood, the noise is fixed at $\sigma^2$ everywhere, and moving the point to the right does not improve the quality of fit of the mean function enough locally to provide a significant gradient. Therefore the points become stuck, and we believe this effect accounts for the failure of the PLV likelihood in Figure 1(c).

It should be emphasised that the *global* optimum of the PLV likelihood (10) may well be a good solution, but it is going to be difficult to find with *gradients*. The SPGP likelihood (9) also suffers from local optima of course, but not so catastrophically. It may be interesting in the future to compare which performs better for hyperparameter optimization.

## 4    Experiments

In the previous section we showed our gradient method successfully learning the pseudo-inputs on a 1D example. There the initial pseudo input points were chosen adversarially, but on a real problem it is sensible to initialize by randomly placing them on real data points,

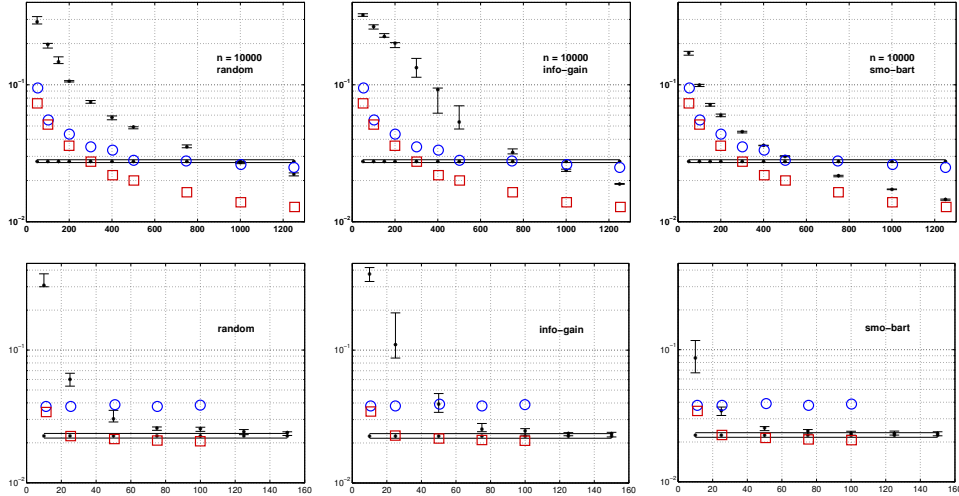

Figure 3: Our results have been added to plots reproduced with kind permission from [7]. The plots show mean square test error as a function of active/pseudo set size $M$. Top row – data set *kin-40k*, bottom row – *pumadyn-32nm*[1]. We have added circles which show SPGP with both hyperparameter and pseudo-input learning from random initialisation. For *kin-40k* the squares show SPGP with hyperparameters obtained from a full GP and fixed. For *pumadyn-32nm* the squares show hyperparameters *initialized* from a full GP. *random*, *info-gain* and *smo-bart* are explained in the text. The horizontal lines are a full GP trained on a subset of the data.

and this is what we do for all of our experiments. To compare our results to other methods we have run experiments on exactly the same data sets as in Seeger et al. [7], following precisely their preprocessing and testing methods. In Figure 3, we have reproduced their learning curves for two large data sets[1], superimposing our test error (mean squared).

Seeger et al. compare three methods: *random*, *info-gain* and *smo-bart*. *random* involves picking an active set of size $M$ randomly from among training data. *info-gain* is their own greedy subset selection method, which is extremely cheap to train – barely more expensive than *random*. *smo-bart* is Smola and Bartlett's [1] more expensive greedy subset selection method. Also shown with horizontal lines is the test error for a full GP trained on a subset of the data of size 2000 for data set *kin-40k* and 1024 for *pumadyn-32nm*. For these learning curves, they do *not* actually learn hyperparameters by maximizing their approximation to the marginal likelihood (10). Instead they fix them to those obtained from the full GP[2].

For *kin-40k* we follow Seeger et al.'s procedure of setting the hyperparameters from the full GP on a subset. We then optimize the pseudo-input positions, and plot the results as red squares. We see the SPGP learning curve lying significantly below all three other methods in Figure 3. We rapidly approach the error of a full GP trained on 2000 points, using a pseudo set of only a few hundred points. We then try the harder task of also finding the hyperparameters at the same time as the pseudo-inputs. The results are plotted as blue circles. The method performs extremely well for small $M$, but we see some overfitting

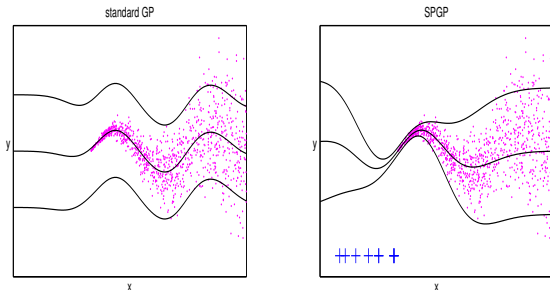

Figure 4: Regression on a data set with input dependent noise. Left: standard GP. Right: SPGP. Predictive mean and two standard deviation lines are shown. Crosses show final locations of pseudo-inputs for SPGP. Hyperparameters are also learnt.

behaviour for large $M$ which seems to be caused by the noise hyperparameter being driven too small (the blue circles have higher likelihood than the red squares below them).

For data set *pumadyn-32nm*, we again try to jointly find hyperparameters and pseudo-inputs. Again Figure 3 shows SPGP with extremely low error for small pseudo set size – with just 10 pseudo-inputs we are already close to the error of a full GP trained on 1024 points. However, in this case increasing the pseudo set size does not decrease our error. In this problem there is a large number of irrelevant attributes, and the relevant ones need to be singled out by ARD. Although the hyperparameters learnt by our method are reasonable (2 out of the 4 relevant dimensions are found), they are not good enough to get down to the error of the full GP. However if we initialize our gradient algorithm with the hyperparameters of the full GP, we get the points plotted as squares (this time red likelihoods > blue likelihoods, so it is a problem of local optima not overfitting). Now with only a pseudo set of size 25 we reach the performance of the full GP, and significantly outperform the other methods (which also had their hyperparameters set from the full GP).

Another main difference between the methods lies in training time. Our method performs optimization over a potentially large parameter space, and hence is relatively expensive to train. On the face of it methods such as *info-gain* and *random* are extremely cheap. However all these methods must be combined with obtaining hyperparameters in some way – either by a full GP on a subset (generally expensive), or by gradient ascent on an approximation to the likelihood. When you consider this combined task, and that all methods involve some kind of gradient based procedure, then none of the methods are particularly cheap. We believe that the gain in accuracy achieved by our method can often be worth the extra training time associated with optimizing in a larger parameter space.

## 5   Conclusions, extensions and future work

Although GPs are very flexible regression models, they are still limited by the form of the covariance function. For example it is difficult to model non-stationary processes with a GP because it is hard to construct sensible non-stationary covariance functions. Although the SPGP is not specifically designed to model non-stationarity, the extra flexibility associated with moving pseudo inputs around can actually achieve this to a certain extent. Figure 4 shows the SPGP fit to some data with an input dependent noise variance. The SPGP achieves a much better fit to the data than the standard GP by moving almost all the pseudo-input points outside the region of data[3]. It will be interesting to test these capabilities further in the future. The extension to classification is also a natural avenue to explore.

We have demonstrated a significant decrease in test error over the other methods for a given small pseudo/active set size. Our method runs into problems when we consider much larger

pseudo set size and/or high dimensional input spaces, because the space in which we are optimizing becomes impractically big. However we have currently only tried using an 'off the shelf' conjugate gradient minimizer, or L-BFGS, and there are certainly improvements that can be made in this area. For example we can try optimizing subsets of variables iteratively (chunking), or stochastic gradient ascent, or we could make a hybrid by picking some points randomly and optimizing others. In general though we consider our method most useful when one wants a very sparse (hence fast prediction) and accurate solution. One further way in which to deal with large $D$ is to learn a low dimensional projection of the input space. This has been considered for GPs before [14], and could easily be applied to our model.

In conclusion, we have presented a new method for sparse GP regression, which shows a significant performance gain over other methods especially when searching for an extremely sparse solution. We have shown that the added flexibility of moving pseudo-input points which are not constrained to lie on the true data points leads to better solutions, and even some non-stationary effects can be modelled. Finally we have shown that hyperparameters can be jointly learned with pseudo-input points with reasonable success.

### Acknowledgements

Thanks to the authors of [7] for agreeing to make their results and plots available for reproduction. Thanks to all at the Sheffield GP workshop for helping to clarify this work.

## Footnotes

[1]*kin-40k*: 10000 training, 30000 test, 9 attributes, see *www.igi.tugraz.at/aschwaig/data.html*. *pumadyn-32nm*: 7168 training, 1024 test, 33 attributes, see *www.cs.toronto/ delve*.

[2]Seeger et al. have a separate section testing their likelihood approximation (10) to learn hyperparameters, in conjunction with the active set selection methods. They show that it can be used to reliably learn hyperparameters with *info-gain* for active set sizes of 100 and above. They have more trouble reliably learning hyperparameters for very small active sets.

[3]It should be said that there are local optima in this problem, and other solutions looked closer to the standard GP. We ran the method 5 times with random initialisations. All runs had higher likelihood than the GP; the one with the highest likelihood is plotted.

### References

[1] A. J. Smola and P. Bartlett. Sparse greedy Gaussian process regression. In *Advances in Neural Information Processing Systems 13*. MIT Press, 2000.

[2] C. K. I. Williams and M. Seeger. Using the Nyström method to speed up kernel machines. In *Advances in Neural Information Processing Systems 13*. MIT Press, 2000.

[3] V. Tresp. A Bayesian committee machine. *Neural Computation*, 12:2719–2741, 2000.

[4] L. Csató. Sparse online Gaussian processes. *Neural Computation*, 14:641–668, 2002.

[5] L. Csató. *Gaussian Processes — Iterative Sparse Approximations*. PhD thesis, Aston University, UK, 2002.

[6] N. D. Lawrence, M. Seeger, and R. Herbrich. Fast sparse Gaussian process methods: the informative vector machine. In *Advances in Neural Information Processing Systems 15*. MIT Press, 2002.

[7] M. Seeger, C. K. I. Williams, and N. D. Lawrence. Fast forward selection to speed up sparse Gaussian process regression. In C. M. Bishop and B. J. Frey, editors, *Proceedings of the Ninth International Workshop on Artificial Intelligence and Statistics*, 2003.

[8] M. Seeger. *Bayesian Gaussian Process Models: PAC-Bayesian Generalisation Error Bounds and Sparse Approximations*. PhD thesis, University of Edinburgh, 2003.

[9] J. Quiñonero Candela. *Learning with Uncertainty — Gaussian Processes and Relevance Vector Machines*. PhD thesis, Technical University of Denmark, 2004.

[10] D. J. C. MacKay. Introduction to Gaussian processes. In C. M. Bishop, editor, *Neural Networks and Machine Learning*, NATO ASI Series, pages 133–166. Kluwer Academic Press, 1998.

[11] C. K. I. Williams and C. E. Rasmussen. Gaussian processes for regression. In *Advances in Neural Information Processing Systems 8*. MIT Press, 1996.

[12] C. E. Rasmussen. *Evaluation of Gaussian Processes and Other Methods for Non-Linear Regression*. PhD thesis, University of Toronto, 1996.

[13] M. N. Gibbs. *Bayesian Gaussian Processes for Regression and Classification*. PhD thesis, Cambridge University, 1997.

[14] F. Vivarelli and C. K. I. Williams. Discovering hidden features with Gaussian processes regression. In *Advances in Neural Information Processing Systems 11*. MIT Press, 1998.
